# A Bayes Rule for Density Matrices

**Manfred K. Warmuth**[*]
Computer Science Department
University of California at Santa Cruz
manfred@cse.ucsc.edu

## Abstract

The classical Bayes rule computes the posterior model probability from the prior probability and the data likelihood. We generalize this rule to the case when the prior is a density matrix (symmetric positive definite and trace one) and the data likelihood a covariance matrix. The classical Bayes rule is retained as the special case when the matrices are diagonal.

In the classical setting, the calculation of the probability of the data is an *expected likelihood*, where the expectation is over the prior distribution. In the generalized setting, this is replaced by an *expected variance* calculation where the variance is computed along the eigenvectors of the prior density matrix and the expectation is over the eigenvalues of the density matrix (which form a probability vector). The variances along any direction is determined by the covariance matrix. Curiously enough this expected variance calculation is a quantum measurement where the co-variance matrix specifies the instrument and the prior density matrix the mixture state of the particle. We motivate both the classical and the generalized Bayes rule with a minimum relative entropy principle, where the Kullbach-Leibler version gives the classical Bayes rule and Umegaki's quantum relative entropy the new Bayes rule for density matrices.

## 1 Introduction

In [TRW05] various on-line updates were generalized from vector parameters to matrix parameters. Following [KW97], the updates were derived by minimizing the loss plus a divergence to the last parameter. In this paper we use the same method for deriving a Bayes rule for density matrices (symmetric positive definite matrices of trace one). When the parameters are probability vectors over the set of models, then the "classical" Bayes rule can be derived using the relative entropy as the divergence (e.g.[KW99, SWRL03]). Analogously we now use the quantum relative entropy, introduced by Umegaki, to derive the generalized Bayes rule.

---
[*]Supported by NSF grant CCR 9821087. Some of this work was done while visiting National ICT Australia in Canberra

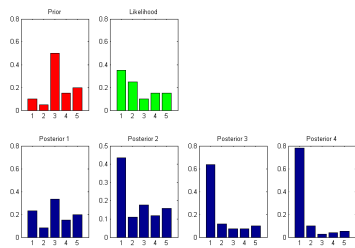

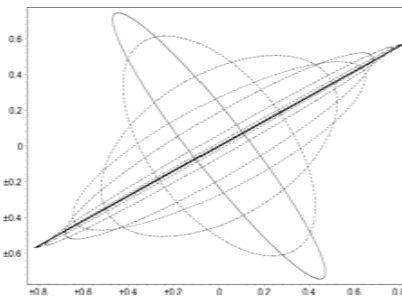

Figure 1: We update the prior four times based on the same data likelihood vector $P(y|M_i)$. The initial posteriors are close to the prior but eventually the posteriors focus their weight on $\text{argmax}_i P(y|M_i)$. The classical Bayes rule may be seen as a soft maximum calculation.

Figure 2: We depict seven iterations of the generalized Bayes rule with the bold NW-SE ellipse as the prior density and the bold-dashed SE-NW ellipse as data covariance matrix. The posterior density matrices (dashed) gradually move from the prior to the longest axis of the covariance matrix.

The new rule uses matrix logarithms and exponentials to avoid the fact that symmetric positive definite matrices are not closed under the matrix product. The rule is strikingly similar to the classical Bayes rule and retains the latter as a special case when the matrices are diagonal. Various cancellations occur when the classical Bayes rule is applied iteratively and similar cancellations happen with the new rule. We shall see that the classical Bayes rule may be seen a soft maximum calculation and the new rule as a soft calculation of the eigenvector with the largest eigenvalue (See figures 1 and 2).

The mathematics applied in this paper is most commonly used in quantum physics. For example, the data likelihood becomes a quantum measurement. It is tempting to call the new rule the "quantum Bayes rule". However, we have no physical interpretation of the this rule. The measurement does not collapse our state and we don't use the unitary evolution of a state to model the rule. Also, the term "quantum Bayes rule" has been claimed before in [SBC01] where the classical Bayes rule is used to update probabilities that happen to arise in the context of quantum physics. In contrast, in this paper our parameters are density matrices.

Our work is most closely related to a paper by Cerf and Adam [CA99] who also give a formula for conditional densities that relies on the matrix exponential and logarithm. However they are interested in the multivariate case (which requires the use of tensors) and their motivation is to obtain a generalization of a conditional quantum entropy. We hope to build on the great body of work done with the classical Bayes rule in the statistics community and therefore believe that this line of research holds great promise.

## 2   The Classical Bayes Rule

To establish a common notation we begin by introducing the familiar Bayes rule. Assume we have $n$ models $M_1, \ldots, M_n$. In the classical setup, model $M_i$ is chosen with prior probability $P(M_i)$ and then $M_i$ generates a datum $y$ with probability $P(y|M_i)$. After observing $y$, the *posterior* probabilities of model $M_i$ are calculated via *Bayes Rule*:

$$P(M_i|y) \quad = \quad \frac{P(M_i)P(y|M_i)}{\sum_j P(M_j)P(y|M_j)}. \tag{1}$$

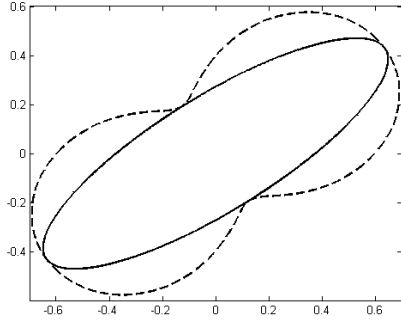
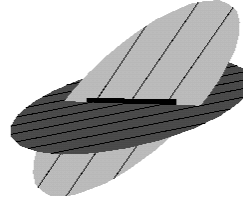

Figure 3: An ellipse $\boldsymbol{S}$ in $\mathbb{R}^2$: The eigenvectors are the directions of the axes and the eigenvalues their lengths. Ellipses are weighted combinations of the one-dimensional degenerate ellipses (dyads) corresponding to the axes. (For unit $\boldsymbol{u}$, the dyad $\boldsymbol{u}\boldsymbol{u}^\top$ is a degenerate one-dimensional ellipse with its single axis in direction $\boldsymbol{u}$.) The solid curve of the ellipse is a plot of $\boldsymbol{S}\boldsymbol{u}$ and the outer dashed figure eight is direction $\boldsymbol{u}$ times the variance $\boldsymbol{u}^\top \boldsymbol{S}\boldsymbol{u}$. At the eigenvectors, this variance equals the eigenvalues and touches the ellipse.

Figure 4: When the ellipse $\boldsymbol{S}$ and $\boldsymbol{T}$ don't have the same span, then $\boldsymbol{S} \odot \boldsymbol{T}$ lies in the intersection of both spans and is a degenerate ellipse of dimension one (bold line). This generalizes the following intersection property of the matrix product when $\boldsymbol{S}$ and $\boldsymbol{T}$ are both diagonal (here of dimension four):

| d$iag(\boldsymbol{S})$ | d$iag(\boldsymbol{T})$ | d$iag(\boldsymbol{ST})$ |
|:---:|:---:|:---:|
| 0 | 0 | 0 |
| a | 0 | 0 |
| 0 | b | 0 |
| a | b | ab |

.

See Figure 1 for a bar plot of the effect of the update on the posterior. By the Theorem of Total Probability, the expected likelihood in the denominator equals $P(y)$. In a moment we will replace this expected likelihood by an expected variance.

## 3    Density Matrices as Priors

We now let our prior $\boldsymbol{D}$ be an arbitrary symmetric positive[1] definite matrix of trace one. Such matrices are called *density matrices* in quantum physics. An outer product $\boldsymbol{u}\boldsymbol{u}^T$, where $\boldsymbol{u}$ has unit length is called a *dyad*. Any *mixture* $\sum_i \alpha_i \, \boldsymbol{a}_i\boldsymbol{a}_i^\top$ of dyads $\boldsymbol{a}_i\boldsymbol{a}_i^\top$ is a density matrix as long as the coefficients $\alpha_i$ are non-negative and sum to one. This is true even if the number of dyads is larger or smaller than the dimension of $\boldsymbol{D}$. The trace of such a mixture is one because dyads have trace one and $\sum_i \alpha_i = 1$. Of course any density matrix $\boldsymbol{D}$ can be decomposed based on an eigensystem. That is, $\boldsymbol{D} = \boldsymbol{D}\boldsymbol{\delta}\boldsymbol{D}^\top$ where $\boldsymbol{D}\boldsymbol{D}^\top = \boldsymbol{I}$. Now the vector of eigenvalues $(\delta_i)$ forms a probability vector equal to the dimension of the density.

In quantum physics, the dyads are called *pure states* and density matrices are mixtures over such states. Note that in this paper we want to address the statistics community and use linear algebra notation instead of Dirac notation. The probability vector $(P(M_i))$ can be represented as a diagonal matrix d$iag((P(M_i))) = \sum_i P(M_i) \, \boldsymbol{e}_i\boldsymbol{e}_i^\top$, where $\boldsymbol{e}_i$ denotes the $i$th standard basis vector. This means that

probability vectors are special density matrices where the eigenvectors are fixed to the standard basis vectors.

## 4 Co-variance Matrices and Basic Notation

In this paper we replace the (conditional) data likelihoods $P(y|M_i)$ by a data covariance matrix $\boldsymbol{\mathcal{D}}(y|.)$ (symmetric positive definite matrix). We now discuss such matrices in more detail.

A covariance matrix $\boldsymbol{\mathcal{S}}$ can be depicted as an ellipse $\{\boldsymbol{\mathcal{S}}\boldsymbol{u} : ||\boldsymbol{u}||_2 \leq 1\}$ centered at the origin, where the eigenvectors form the principal axes and the eigenvalues are the lengths of the axes (See Figure 3). Assume $\boldsymbol{\mathcal{S}}$ is the covariance matrix of some random cost vector $\boldsymbol{c} \in \mathbb{R}^n$, i.e. $\boldsymbol{\mathcal{S}} = \mathbb{E}\left((\boldsymbol{c} - \mathbb{E}(\boldsymbol{c})(\boldsymbol{c} - \mathbb{E}(\boldsymbol{c}))^\top\right)$. Note that a covariance matrix $\boldsymbol{\mathcal{S}}$ is diagonal if the components of the cost vector are independent. The variance of the cost vector $\boldsymbol{c}$ along a unit vector $\boldsymbol{u}$ has the form

$$\mathbb{V}(\boldsymbol{c}^\top \boldsymbol{u}) = \mathbb{E}\left(\left(\boldsymbol{c}^\top \boldsymbol{u} - \mathbb{E}(\boldsymbol{c}^\top \boldsymbol{u})\right)^2\right) = \mathbb{E}\left(\left((\boldsymbol{c}^\top - \mathbb{E}(\boldsymbol{c}^\top))\,\boldsymbol{u}\right)^2\right) = \boldsymbol{u}^\top \boldsymbol{\mathcal{S}} \boldsymbol{u}$$

and the variance along an eigenvector is the corresponding eigenvalue (See Figure 3). Using this interpretation, the matrix $\boldsymbol{\mathcal{S}}$ may be seen as a mapping $\boldsymbol{\mathcal{S}}(.)$ from the unit ball to $\mathbb{R}_{\geq 0}$, i.e. $\boldsymbol{\mathcal{S}}(\boldsymbol{u}) = \boldsymbol{u}^\top \boldsymbol{\mathcal{S}} \boldsymbol{u}$.

A second interpretation of the scalar $\boldsymbol{u}^\top \boldsymbol{\mathcal{S}} \boldsymbol{u}$ is the square length of $\boldsymbol{u}$ w.r.t. the basis $\sqrt{\boldsymbol{\mathcal{S}}}$, that is $\boldsymbol{u}^\top \boldsymbol{\mathcal{S}} \boldsymbol{u} = \boldsymbol{u}^\top \sqrt{\boldsymbol{\mathcal{S}}}\sqrt{\boldsymbol{\mathcal{S}}}\boldsymbol{u} = ||\sqrt{\boldsymbol{\mathcal{S}}}\boldsymbol{u}||_2^2$. Thirdly, $\boldsymbol{u}^T \boldsymbol{\mathcal{S}} \boldsymbol{u}$ is a *quantum measurement* of the pure state $\boldsymbol{u}$ with an instrument represented by $\boldsymbol{\mathcal{S}}$. Since the square length of $\boldsymbol{u}$ w.r.t. any orthogonal basis $\boldsymbol{S}$ is one, any such basis turns the unit vector into an $n$-dimensional probability vector $((\boldsymbol{u}^\top \boldsymbol{s}_i)^2)$. Now $\boldsymbol{u}^\top \boldsymbol{\mathcal{S}} \boldsymbol{u}$ is the expected eigenvalue w.r.t. this probability vector: $\boldsymbol{u}^\top \boldsymbol{\mathcal{S}} \boldsymbol{u} = \sum_i \sigma_i (\boldsymbol{u}^\top \boldsymbol{s}_i)^2$.

The trace $\mathrm{tr}(\boldsymbol{A})$ of a square matrix $\boldsymbol{A}$ is the sum of its diagonal elements $\boldsymbol{A}_{ii}$. Recall that $\mathrm{tr}(\boldsymbol{A}\boldsymbol{B}) = \mathrm{tr}(\boldsymbol{B}\boldsymbol{A})$ for any matrices $\boldsymbol{A} \in \mathbb{R}^{n \times m}$, $\boldsymbol{B} \in \mathbb{R}^{m \times n}$. The trace is *unitarily invariant*, i.e. for any orthogonal matrix $\boldsymbol{U}$, $\mathrm{tr}(\boldsymbol{U}\boldsymbol{A}\boldsymbol{U}^\top) = \mathrm{tr}(\boldsymbol{U}^\top \boldsymbol{U}\boldsymbol{A}) = \mathrm{tr}(\boldsymbol{A})$. Also, $\mathrm{tr}(\boldsymbol{u}\boldsymbol{u}^\top \boldsymbol{A}) = \mathrm{tr}(\boldsymbol{u}^\top \boldsymbol{A}\boldsymbol{u}) = \boldsymbol{u}^\top \boldsymbol{A}\boldsymbol{u}$. Therefore the trace of a square matrix may be seen as the total variance along any set of orthogonal directions:

$$\mathrm{tr}(\boldsymbol{A}) = \mathrm{tr}(\boldsymbol{I}\boldsymbol{A}) = \mathrm{tr}(\sum_i \boldsymbol{u}_i \boldsymbol{u}_i^\top \boldsymbol{A}) = \sum_i \boldsymbol{u}_i^\top \boldsymbol{A}\boldsymbol{u}_i.$$

In particular, the trace of a square matrix is the sum of its eigenvalues.

The matrix exponential $\exp(\boldsymbol{\mathcal{S}})$ of the symmetric matrix $\boldsymbol{\mathcal{S}} = \boldsymbol{S}\boldsymbol{\sigma}\boldsymbol{S}^\top$ is defined as $\boldsymbol{S}\exp(\boldsymbol{\sigma})\boldsymbol{S}^\top$, where $\exp(\boldsymbol{\sigma})$ is obtained by exponentiating the diagonal entries (eigenvalues). The matrix logarithm $\log(\boldsymbol{\mathcal{S}})$ is defined similarly but now $\boldsymbol{\mathcal{S}}$ must be strictly positive definite. Clearly, the two functions are inverses of each other. It is important to remember that $\exp(\boldsymbol{\mathcal{S}} + \boldsymbol{\mathcal{T}}) = \exp(\boldsymbol{\mathcal{S}})\exp(\boldsymbol{\mathcal{T}})$ only holds iff the two symmetric matrices commute[2], i.e. $\boldsymbol{\mathcal{S}}\boldsymbol{\mathcal{T}} = \boldsymbol{\mathcal{T}}\boldsymbol{\mathcal{S}}$. However, the following trace inequality, known as the Golden-Thompson inequality [Bha97], always holds:

$$\mathrm{tr}(\exp\boldsymbol{\mathcal{S}}\exp\boldsymbol{\mathcal{T}}) \geq \mathrm{tr}(\exp(\boldsymbol{\mathcal{S}} + \boldsymbol{\mathcal{T}})). \tag{2}$$

## 5 The Generalized Bayes Rule

The following experiment underlies the more general setup: If the prior is $\boldsymbol{\mathcal{D}}(.) = \sum_i \delta_i \, \boldsymbol{d}_i \boldsymbol{d}_i^\top$, then the dyad (or pure state) $\boldsymbol{d}_i \boldsymbol{d}_i^\top$ is chosen with probability $\delta_i$ and a random variable $\boldsymbol{c}^\top \boldsymbol{d}_i$ is observed where $\boldsymbol{c}$ has covariance matrix $\boldsymbol{\mathcal{D}}(y|.)$.

In our generalization we replace the expected data likelihood $P(y) = \sum_i P(M_i)P(y|M_i)$ by the following trace:

$$\text{tr}(\boldsymbol{\mathcal{D}}(.)\boldsymbol{\mathcal{D}}(y|.)) = \text{tr}(\sum_i \delta_i \, \boldsymbol{d}_i \boldsymbol{d}_i{}^\top \boldsymbol{\mathcal{D}}(y|.)) = \sum_i \delta_i \, \boldsymbol{d}_i{}^\top \boldsymbol{\mathcal{D}}(y|.)\boldsymbol{d}_i.$$

Recall that $\boldsymbol{d}_i{}^\top \boldsymbol{\mathcal{D}}(y|.)\boldsymbol{d}_i$ is the variance of $\boldsymbol{c}$ in direction $\boldsymbol{d}_i$: i.e. $\mathbb{V}(\boldsymbol{c}^\top \boldsymbol{d}_i)$. Therefore the above trace is the expected variance along the eigenvectors of the density matrix weighted by the eigenvalues. Curiously enough, this trace computation is a *quantum measurement*, where $\boldsymbol{\mathcal{D}}(y|.)$ represents the instrument and $\boldsymbol{\mathcal{D}}(.)$ the mixture state of the particle.

In the generalized Bayes rule we cannot simply multiply the prior density matrix with the covariance matrix that corresponds to the data likelihood. This is because a product of two symmetric positive definite matrices may be neither symmetric nor positive definite. Instead we define the operation $\odot$ on the cone of symmetric positive definite matrices. We begin by defining this operation for the case when the matrices $\boldsymbol{\mathcal{S}}$ and $\boldsymbol{\mathcal{T}}$ are strictly positive definite (and symmetric):

$$\boldsymbol{\mathcal{S}} \odot \boldsymbol{\mathcal{T}} := \exp(\log \boldsymbol{\mathcal{S}} + \log \boldsymbol{\mathcal{T}}). \tag{3}$$

The matrix log of both matrices produces symmetric matrices that sum to a symmetric matrix. Finally the matrix exponential of the sum produces again a symmetric positive matrix. Note that the matrix log is not defined when the matrix has a zero eigenvalue. However for arbitrary symmetric positive definite matrices one can define the operation $\odot$ as the following limit:

$$\boldsymbol{\mathcal{S}} \odot \boldsymbol{\mathcal{T}} := \lim_{n \to \infty} (\boldsymbol{\mathcal{S}}^{1/n} \boldsymbol{\mathcal{T}}^{1/n})^n.$$

This limit is the *Lie Product Formula* [Bha97] when $\boldsymbol{\mathcal{S}}$ and $\boldsymbol{\mathcal{T}}$ are both strictly positive, but it exists even if the matrices don't have full rank and by Theorem 1.2 of [Sim79],

$$range(\boldsymbol{\mathcal{S}} \odot \boldsymbol{\mathcal{T}}) = range(\boldsymbol{\mathcal{S}}) \cap range(\boldsymbol{\mathcal{T}}).$$

Assume that $k$ is the dimension of $range(\boldsymbol{\mathcal{S}}) \cap range(\boldsymbol{\mathcal{T}})$, that $\boldsymbol{B}$ is an orthonormal basis of $range(\boldsymbol{\mathcal{S}}) \cap range(\boldsymbol{\mathcal{T}})$ (i.e. $\boldsymbol{B} \in \mathbb{R}^{n \times k}$, $\boldsymbol{B}^T \boldsymbol{B} = \boldsymbol{I}_k$, and $range(\boldsymbol{B}) = range(\boldsymbol{\mathcal{S}}) \cap range(\boldsymbol{\mathcal{T}})$) and that $\log^+$ denotes the modified matrix logarithm that takes logs of the non-zero eigenvalues but leaves zero eigenvalues unchanged. Then by the same theorem[3],

$$\boldsymbol{\mathcal{S}} \odot \boldsymbol{\mathcal{T}} = \boldsymbol{B} \, \exp(\boldsymbol{B}^T (\log^+ \boldsymbol{\mathcal{S}} + \log^+ \boldsymbol{\mathcal{T}})\boldsymbol{B}) \, \boldsymbol{B}^T. \tag{4}$$

When both matrices have the same eigensystem, then $\odot$ becomes the matrix product. One can show that $\odot$ is associative, commutative, has the identity matrix $\boldsymbol{I}$ as its neutral element and for any strictly positive definite and symmetric matrix $\boldsymbol{\mathcal{S}}$, $\boldsymbol{\mathcal{S}} \odot \boldsymbol{\mathcal{S}}^{-1} = \boldsymbol{I}$. Finally, $(c\boldsymbol{\mathcal{S}}) \odot \boldsymbol{\mathcal{T}} = c(\boldsymbol{\mathcal{S}} \odot \boldsymbol{\mathcal{T}})$, for any non-negative scalar.

Using this new product operation, the generalized Bayes rule becomes:

$$\boldsymbol{\mathcal{D}}(.|y) = \frac{\boldsymbol{\mathcal{D}}(.) \odot \boldsymbol{\mathcal{D}}(y|.)}{\text{tr}(\boldsymbol{\mathcal{D}}(.) \odot \boldsymbol{\mathcal{D}}(y|.))}. \tag{5}$$

Normalizing by the trace assures that the trace of the posterior density matrix is one. As we see in Figure 2, this posterior moves toward the largest axis of the data covariance matrix and the new rule can be interpreted as a soft calculation of the

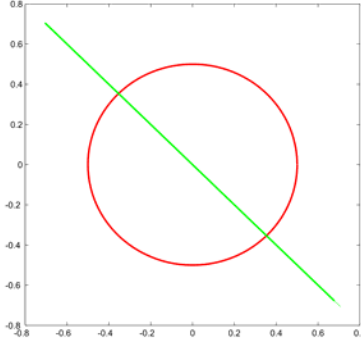

Figure 5: Assume the prior density matrix is the circle $\boldsymbol{D}(.) = \begin{pmatrix} \frac{1}{2} & 0 \\ 0 & \frac{1}{2} \end{pmatrix}$ and the data covariance matrix the degenerate NE-SW ellipse $\boldsymbol{D}(y|.) = \frac{1}{2}\begin{pmatrix} 1 & -1 \\ -1 & 1 \end{pmatrix} = \boldsymbol{U}\begin{pmatrix} 0 & 0 \\ 0 & 1 \end{pmatrix}\boldsymbol{U}^\top$, where $\boldsymbol{U} = \begin{pmatrix} \frac{1}{\sqrt{2}} & \frac{1}{\sqrt{2}} \\ \frac{1}{\sqrt{2}} & -\frac{1}{\sqrt{2}} \end{pmatrix}$. Now for all diagonal matrices $\boldsymbol{S}(.)$, $\mathrm{tr}(\boldsymbol{S}(.)\,\boldsymbol{D}(y|.)) = \frac{1}{2}$, i.e. largest eigenvalue is not "visible" in basis $\boldsymbol{I}$. But $\mathrm{tr}\left(\underbrace{U\,(\begin{smallmatrix}0&0\\0&1\end{smallmatrix})\,\boldsymbol{U}^\top}_{\boldsymbol{D}(.|y)\text{ of new rule}}\quad \boldsymbol{D}(y|.)\right) = 1$.

eigenvector with maximum eigenvalue. When the matrices $\boldsymbol{D}(.)$ and $\boldsymbol{D}(y|.)$ have the same eigensystem, then $\odot$ becomes the matrix multiplication. In particular, when the prior is $diag((P(M_i)))$ and the covariance matrix $diag((P(y|M_i))$, then the new rule realizes the classical rule and computes $diag((P(M_i|y))$. Figure 5 gives an example that shows how the off-diagonal elements can be exploited by the new rule.

In the classical Bayes rule, the normalization factor is the expected data likelihood. In the case of the generalized Bayes rule, the expected variance only upper bounds the normalization factor via the Golden-Thompsen inequality (2):
$$\mathrm{tr}(\boldsymbol{D}(.)\boldsymbol{D}(y|.)) \geq \mathrm{tr}(\boldsymbol{D}(.) \odot \boldsymbol{D}(y|.)). \tag{6}$$

The classical Bayes rule can be applied iteratively to a sequence of data and various cancellations occur. For the sake of simplicity we only consider two data points $y_1$, $y_2$:
$$P(M_i|y_2y_1) = \frac{P(M_i|y_1)P(y_2|M_i,y_1)}{P(y_2|y_1)} = \frac{P(M_i)P(y_1|M_i)P(y_2|M_i,y_1)}{P(y_2y_1)}.$$

$$P(y_2|y_1)P(y_1) = (\sum_i \underbrace{P(M_i|y_1)}_{use(1)} P(y_2|M_i,y_1))(\sum_i P(M_i)P(y_1|M_i))$$
$$= \sum_i P(M_i)P(y_1|M_i)P(y_2|M_i,y_1) = P(y_2y_1). \tag{7}$$

Analogously,
$$\boldsymbol{D}(.|y_2y_1) = \frac{\boldsymbol{D}(.|y_1) \odot \boldsymbol{D}(y_2|.,y_1)}{\mathrm{tr}(\boldsymbol{D}(.|y_1) \odot \boldsymbol{D}(y_2|.,y_1))} = \frac{\boldsymbol{D}(.) \odot \boldsymbol{D}(y_1|.) \odot \boldsymbol{D}(y_2|.,y_1)}{\mathrm{tr}(\boldsymbol{D}(.) \odot \boldsymbol{D}(y_1|.) \odot \boldsymbol{D}(y_2|.,y_1))}.$$

Finally, the product of the expected variance for both trials combine in a similar way, except that in the generalized case the equality becomes an inequality:

$$\text{tr}(\boldsymbol{\mathcal{D}}(.|y_1)\boldsymbol{\mathcal{D}}(y_2|.,y_1))\, \text{tr}(\boldsymbol{\mathcal{D}}(.)\boldsymbol{\mathcal{D}}(y_1|.))$$

$$\geq \quad \text{tr}(\underbrace{\boldsymbol{\mathcal{D}}(.|y_1)}_{use(5)} \odot \boldsymbol{\mathcal{D}}(y_2|.,y_1))\, \text{tr}(\boldsymbol{\mathcal{D}}(.) \odot \boldsymbol{\mathcal{D}}(y_1|.))$$

$$= \quad -\log \text{tr}(\boldsymbol{\mathcal{D}}(.) \odot \boldsymbol{\mathcal{D}}(y_1|.) \odot \boldsymbol{\mathcal{D}}(y_2|.,y_1)).$$

The above inequality is an instantiation of the Golden-Thompsen inequality (2) and the above equality generalizes the middle equality in (7).

## 6 The Derivation of the Generalized Bayes Rule

The classical Bayes rule can be derived[4] by minimizing a relative entropy to the prior plus a convex combination of the log losses of the models (See e.g. [KW99, SWRL03]):

$$\inf_{\gamma_i \geq 0,\, \sum_i \gamma_i = 1} \sum_i \gamma_i \ln \frac{\gamma_i}{P(M_i)} \; - \; \sum_i \gamma_i \log P(y|M_i).$$

Without the relative entropy, the argument of the infimum is linear in the weights $\gamma_i$ and is minimized when all weight is placed on the maximum likelihood models, i.e. the set of indices $\text{argmax}_i P(y|M_i)$. The negative entropy ameliorates the maximum calculation and pulls the optimal solution towards the prior. Observe that the non-negativity constraints can be dropped since the entropy acts as a barrier. By introducing a Lagrange multiplier for the remaining constraint and differentiating, we obtain the solution $\gamma_i^* = \frac{P(M_i)P(y|M_i)}{\sum_j P(M_j)P(y|M_j)}$, which is the classical Bayes rule (1). By plugging $\gamma_i^*$ into the argument of the infimum we obtain the optimum value $-\ln P(y)$. Notice that this is minus the logarithm of the normalization of the Bayes rule (1) and is also the log loss associated the standard Bayesian setup.

To derive the new generalized Bayes rule in an analogous way, we use the quantum physics generalizations of the relative entropy between two densities $\boldsymbol{\mathcal{G}}$ and $\boldsymbol{\mathcal{D}}$ (due to Umegaki): $\text{tr}(\boldsymbol{\mathcal{G}}(\log \boldsymbol{\mathcal{G}} - \log \boldsymbol{\mathcal{D}}))$. We also need to replace the mixture of negative log likelihoods by the trace $-\text{tr}(\boldsymbol{\mathcal{G}} \log \boldsymbol{\mathcal{D}}(y|.))$. Now the matrix parameter $\boldsymbol{\mathcal{G}}$ is constrained to be a density matrix and the minimization problem becomes[5] :

$$\inf_{\boldsymbol{\mathcal{G}}\ \text{dens.matr.}} \text{tr}(\boldsymbol{\mathcal{G}}(\log \boldsymbol{\mathcal{G}} - \log \boldsymbol{\mathcal{D}}(.))) \quad - \quad \text{tr}(\boldsymbol{\mathcal{G}} \log \boldsymbol{\mathcal{D}}(y|.))$$

Except for the quantum relative entropy term, the argument of the infimum is again linear in the variable $\boldsymbol{\mathcal{G}}$ and is minimized when $\boldsymbol{\mathcal{G}}$ is a single dyad $\boldsymbol{u}\boldsymbol{u}^\top$, where $\boldsymbol{u}$ is the eigenvector belonging to maximum eigenvalue of the matrix $\log \boldsymbol{\mathcal{D}}(y|.)$. The linear term pulls $\boldsymbol{\mathcal{G}}$ toward a direction of high variance of this matrix, whereas the quantum relative entropy pulls $\boldsymbol{\mathcal{G}}$ toward the prior density matrix. The density matrix constraint requires the eigenvalues of $\boldsymbol{\mathcal{G}}$ to be non-negative and the trace to $\boldsymbol{\mathcal{G}}$ to be one. The entropy works as a barrier for the non-negativity constraints and thus these constraints can be dropped. Again by introducing a Lagrange multiplier for the remaining trace constraint and differentiating (following [TRW05]), we arrive at a formula for the optimum $\boldsymbol{\mathcal{G}}^*$ which coincides with the formula for the $\boldsymbol{\mathcal{D}}(.|y)$ given in the generalized Bayes rule (5), where $\odot$ is defined[6] as in (3). Since the quantum relative entropy is strictly convex [NC00] in $\boldsymbol{\mathcal{G}}$, the optimum $\boldsymbol{\mathcal{G}}^*$ is unique.

# 7 Conclusion

Our generalized Bayes rule suggests a definition of conditional density matrices and we are currently developing a calculus for such matrices. In particular, a common formalism is needed that includes the multivariate conditional density matrices defined in [CA99] based on tensors.

In this paper we only considered real symmetric matrices. However, our methods immediately generalize to complex Hermitian matrices, i.e square matrices in $\mathbb{C}^{n \times n}$ for which $\boldsymbol{\mathcal{S}} = \overline{\boldsymbol{\mathcal{S}}}^T = \boldsymbol{\mathcal{S}}^*$. Now both the prior density matrix and the data covariance matrix must be Hermitian instead of symmetric.

The generalized Bayes rule for symmetric positive definite matrices relies on computing eigendecompositions ($\Omega(n^3)$ time). Hopefully, there exist $O(n^2)$ versions of the update that approximate the generalized Bayes rule sufficiently well.

Extensive research has been done in the so-called "expert framework" (see e.g.[KW99] for a list of references) where a mixture over experts is maintained by the on-line algorithm for the purpose of performing as well as the best expert chosen in hindsight. In preliminary research we showed that one can maintain a density matrix over the base experts instead and derive updates similar to the generalized Bayes rule given in this paper. Most importantly, the bounds generalize to the case when mixtures over experts are replaced by density matrices.

**Acknowledgment:** We would like to thank Dima Kuzmin for his extensive help with all aspects of this paper. Thanks also to Torsten Ehrhardt who first proved to us the range intersection and projection properties of the $\odot$ operation.

## Footnotes

[1]We use the convention that positive definite matrices have non-negative eigenvalues and *strictly* positive definite matrices have positive eigenvalues.

[2]This occurs iff the two symmetric matrices have the same eigensystem.

[3]The $\log^+ \boldsymbol{\mathcal{S}}$ term in the formula can be replaced by $\tilde{\boldsymbol{B}} \log(\tilde{\boldsymbol{B}}^T \boldsymbol{\mathcal{S}} \tilde{\boldsymbol{B}})\tilde{\boldsymbol{B}}^T$, where $\tilde{\boldsymbol{B}}$ is an orthonormal basis of $range(\boldsymbol{\mathcal{S}})$, and similarly for $\log^+ \boldsymbol{\mathcal{T}}$.

[4]For the sake of simplicity assume that for all $i$, $P(M_i)$ and $P(y|M_i)$ are non-negative.

[5]Assume here that $\boldsymbol{\mathcal{D}}(.)$ and $\boldsymbol{\mathcal{D}}(y|.)$ are both strictly positive definite.

[6]With some work, one can also derive the Bayes rule with the fancier $\odot$ operation (4).

# References

[Bha97] R. Bhatia. *Matrix Analysis*. Springer, Berlin, 1997.

[CA99] N. J. Cerf and C. Adam. Quantum extension of conditional probability. *Physical Review A*, 60(2):893–897, August 1999.

[KW97] J. Kivinen and M. K. Warmuth. Additive versus exponentiated gradient updates for linear prediction. *Information and Computation*, 132(1):1–64, January 1997.

[KW99] J. Kivinen and M. K. Warmuth. Averaging expert predictions. In *Computational Learning Theory: 4th European Conference (EuroCOLT '99)*, pages 153–167, Berlin, March 1999. Springer.

[NC00] M.A. Nielsen and I.L. Chuang. *Quantum Computation and Quantum Information*. Cambridge University Press, 2000.

[SBC01] R. Schack, T. A. Brun, and C. M. Caves. Quantum Bayes rule. *Physical Review A*, 64(014305), 2001.

[Sim79] Barry Simon. *Functional Integration and Quantum Physics*. Academic Press, New York, 1979.

[SWRL03] R. Singh, M. K. Warmuth, B. Raj, and P. Lamere. Classificaton with free energy at raised temperatures. In *Proc. of EUROSPEECH 2003*, pages 1773–1776, September 2003.

[TRW05] K. Tsuda, G. Rätsch, and M. K. Warmuth. Matrix exponentiated gradient updates for on-line learning and Bregman projections. *Journal of Machine Learning Research*, 6:995–1018, June 2005.
